# Recognizing Handwritten Digits Using Mixtures of Linear Models

**Geoffrey E Hinton**    **Michael Revow**    **Peter Dayan**
Department of Computer Science, University of Toronto
Toronto, Ontario, Canada M5S 1A4

## Abstract

We construct a mixture of locally linear generative models of a collection of pixel-based images of digits, and use them for recognition. Different models of a given digit are used to capture different styles of writing, and new images are classified by evaluating their log-likelihoods under each model. We use an EM-based algorithm in which the M-step is computationally straightforward principal components analysis (PCA). Incorporating tangent-plane information [12] about expected local deformations only requires adding tangent vectors into the sample covariance matrices for the PCA, and it demonstrably improves performance.

## 1 Introduction

The usual way of using a neural network for digit recognition is to train it to output one of the ten classes. When the training data is limited to N examples equally distributed among the classes, there are only $N \log_2 10$ bits of constraint in the class labels so the number of free parameters that can be allowed in a discriminative neural net model is severely limited. An alternative approach, motivated by density estimation, is to train a separate autoencoder network on examples of each digit class and to recognise digits by seeing which autoencoder network gives the best reconstruction of the data. Because the output of each autoencoder has the same dimensionality as the image, each training example provides many more bits of constraint on the parameters. In the example we describe, 7000 training images are sufficient to fit $384,000$ parameters and the training procedure is fast enough to do the fitting overnight on an R4400-based machine.

Auto-encoders can be viewed in terms of minimum description length descriptions of data in which the input-hidden weights produce a code for a particular case and

the hidden-output weights embody a generative model which turns this code back into a close approximation of the original example [14, 7]. Code costs (under some prior) and reconstruction error (squared error assuming an isotropic Gaussian misfit model) sum to give the overall code length which can be viewed as a lower bound on the log probability density that the autoencoder assigns to the image. This properly places the emphasis on designing appropriate generative models for the data which will generalise well by assigning high log probabilities to new patterns from the same classes.

We apply this idea to recognising handwritten digits from grey-level pixel images using linear auto-encoders. Linear hidden units for autoencoders are barely worse than non-linear ones when squared reconstruction error is used [1], but have the great computational advantage during training that input-hidden and hidden-output weights can be derived from principal components analysis (PCA) of the training data. In effect a PCA encoder approximates the entire N dimensional distribution of the data with a lower dimensional "Gaussian pancake" [13], choosing, for optimal data compression, to retain just a few of the PCs. One could build a single PCA model for each digit – however the many different styles of writing suggest that more than one Gaussian pancake should be used, by dividing the population of a single digit class into a number of sub-classes and approximating each class by its own model. A similar idea for data compression was used by [9] where vector quantization was used to define sub-classes and PCA was performed within each sub-class (see also [2]). We used an iterative method based on the Expectation Maximisation (EM) algorithm [4] to fit mixtures of linear models.

The *reductio* of the local linear approach would have just one training pattern in each model. This approach would amount to a nearest neighbour method for recognition using a Euclidean metric for error, a technique which is known to be infelicitous. One reason for this poor performance is that characters do not only differ in writing styles, but also can undergo simple transformations such as translations and rotations. These give rise to somewhat non-linear changes as measured in pixel space. Nearest neighbour methods were dramatically improved methods by defining a metric in which locally linearised versions of these transformations cost nothing [12]. In their tangent distance method, each training or test point is represented by a h-dimensional linear subspace, where each of these dimensions corresponds to a linear version of one of the transformations, and distances are measured between subspaces rather than points. The local linear autoencoder method can be seen just like this – variations along one of the h principal component directions are free, while variations along the remaining principal directions cost. However, rather than storing and testing each training pattern, the local models summarise the regularities over a number of patterns. This reduces storage and recognition time, and allows the directions of free variation to be averaged over numbers of patterns and also to be determined by the data rather than being pre-specified. *A priori* knowledge that particular transformations are important can be incorporated using a version of the tangent-prop procedure [12], which is equivalent in this case to adding in slightly transformed versions of the patterns. Also reconstruction error could be assessed either at a test pattern, or, more like tangent distance, between its transformation subspace and the models' principal components subspaces.

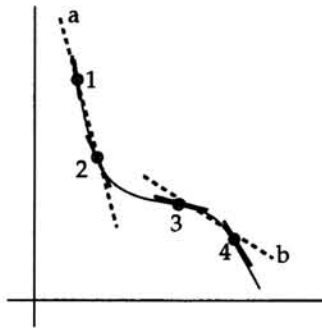

Figure 1: Didactic example of tangent information and local linear models. See text for details.

Figure 1 illustrates the idea. Imagine that the four points 1-4 portray in image space different examples of the same digit, subject to some smooth transformation. As in tangent distance, one could represent this curve using the points and their local tangents (thick lines). However one might do better splitting it into three local linear models rather than four – model 'a' (just a line in this simple case) averages the upper part of the curve more effectively than the combination of the two tangents at '1' and '2'. However, given just the points, one might construct model 'b' for '3' and '4', which would be unfortunate. Incorporating information about the tangents as well would encourage the separation of these segments. Care should be taken in generalising this picture to high dimensional spaces.

The next section develops the theory behind variants of these systems (which is very similar to that in [5, 10]), and section 3 discusses how they perform.

## 2 Theory

Linear auto-encoders embody a model in which variations from the mean of a population along certain directions are cheaper than along others, as measured by the log-unlikelihood of examples. Creating such a generative model is straight-forward. Principal component analysis (PCA) is performed on the training data and the leading $h$ principal components are retained, defining an $h$-dimensional subspace onto which the $n$-dimensional inputs are projected. We choose $h$ using cross-validation, although a minimum description length criterion could also be used. We ignore the effect of different variances along the different principal components and use a model in which the code length for example $i$ (the negative log-likelihood) is proportional to the reconstruction error – the squared Euclidean distance between the output of the autoencoder and the pattern itself.

Rather than having just one autoencoder for each digit, there is a whole collection, and therefore we use EM to assign examples to $n$ sub-classes, just as in clustering using a mixture of Gaussians generative model. During the E-step, the responsibility for each pattern is assigned amongst the sub-classes, and in the M-step PCA is performed, altering the parameters of a sub-class appropriately to minimise the

reconstruction cost of the data for which it is responsible.

Formally, the algorithm is:

1. Choose initial autoencoder assignments for each example in the training set (typically using a K-means clustering algorithm).
2. Perform PCA separately for each autoencoder;
3. Reassign patterns to the autoencoder that reconstructs them the best;
4. Stop if no patterns have changed sub-class, otherwise return to step 2.

There is a 'soft' version of the algorithm in which the responsibility of autoencoder q for example i is calculated as $r_{iq} = e^{-\|E_{iq}\|^2/2\sigma^2}/(\sum_r e^{-\|E_{ir}\|^2/2\sigma^2})$ where $E_{ir}$ is the reconstruction error. For this, in step 2, the examples are weighted for the PCA by the responsibilities, and convergence is assessed by examining the change in the log-likelihood of the data at each iteration. The soft version requires a choice of $\sigma^2$, the assumed variance in the directions orthogonal to the pancake.

The algorithm generates a set of local linear models for each digit. Given a test pattern we evaluate the code length (the log-likelihood) against all the models for all the digits. We use a hard method for classification – determining the identity of the pattern only by the model which reconstructs it best. The absolute quality of the best reconstruction and the relative qualities of slightly sub-optimal reconstructions are available to reject ambiguous cases.

For a given linear model, not counting the code cost implies that deformations of the images along the principal components for the sub-class are free. This is like the metric used by [11] except that they explicitly specified the directions in which deformations should be free, rather than learning them from the data. We wished to incorporate information about the preferred directions without losing the summarisation capacity of the local models, and therefore turned to the tangent prop algorithm [12].

Tangent prop takes into account information about how the output of the system should vary locally with particular distortions of the input by penalising the system for having incorrect derivatives in the relevant directions. In our case, the overall classification process is highly non-linear, making the application of tangent-prop to it computationally unattractive, but it is easy to add a tangent constraint to the reconstruction step because it is linear. Imagine requiring the system $f(p) = A.p$ to reconstruct $x + \lambda t$ and $x - \lambda t$ as well as $x$, for an input example $x$, and distortion $t$ (the tangent vector), and where $\lambda$ is a weight. Their contribution to the error is proportional to $|x - A.x|^2 + \lambda^2|t - A.t|^2$, where the second term is equivalent to the error term that tangent-prop would add. Incorporating this into the PCA is as simple as adding a weighted version of $tt^T$ to the covariance matrix – the tangent vectors are never added to the means of the sub-classes.

## 3  Results

We have evaluated the performance of the system on data from the CEDAR CDROM 1 database containing handwritten digits lifted from mail pieces passing through a United States Post Office [8]. We divided the *br* training set of binary

| Clustering | Recognition | Raw Errors |
|------------|-------------|------------|
| None | None | 62 (3.10%) |
| Heavy | Light | 29 (1.45%) |
| Heavy | None | 45 (2.25%) |
| Heavy | Heavy | 90 (4.50%) |

Table 1: Classification errors on the validation test when different weightings are used for the tangent vectors during clustering and recognition. No rejections were allowed.

segmented digits into 7,000 training examples, 2,000 validation examples and 2,000 "internal test" examples. All digits were equally represented in these sets. The binary images in the database are all of different sizes, so they were scaled onto a 16 × 16 grid and then smoothed with a Gaussian filter.

The validation set was used to investigate different choices for the Gaussian filter variances, the numbers of sub-classes per digit and principal components per model, and the different weightings on the tangent vectors. Clearly this is a large parameter space and we were only able to perform a very coarse search. In the results reported here all digits have the same number of sub-classes (10) and the number of principal components per model was picked so as to explain 95% of the training set variance assigned to that model. Once a reasonable set of parameter settings had been decided upon, we used all 11,000 images to train a final version which was tested on the official *bs* set (2711 images).

There are two major steps to the algorithm; defining sub-classes within each digit and reconstructing for recognition. We found that the tangent vectors should be more heavily weighted in the sub-class clustering step than during ultimate recognition. Figure 2 shows the means of the 10 sub-classes for the digit two where the clustering has been done (a) without or (b) with tangent vectors. It is clear that the clusters defined in (b) capture different styles of 2s in a way that those in (a) do not – they are more diverse and less noisy. They also perform better. The raw error rate (no rejections allowed) on the validation set with different amounts of tangent vector weightings are shown in Table 1. The results on the official test set (2711 examples) are shown in Table 2.

## 4 Discussion and Conclusions

A mixture of local linear models is an effective way to capture the underlying styles of handwritten digits. The first few principal components (less than 20 on 256-dimensional data) extract a significant proportion of the variance in the images within each sub-class. The resulting models classify surprisingly well without requiring large amounts of either storage or computation during recognition. Further, it is computationally easy and demonstrably useful to incorporate tangent information requiring reconstruction to be good for small transformations of the sample patterns in a handful of pre-specified directions. Adding tangent information is exactly equivalent to replacing each digit by a Gaussian cloud of digits perturbed along the tangent plane and is much more efficient than adding extra

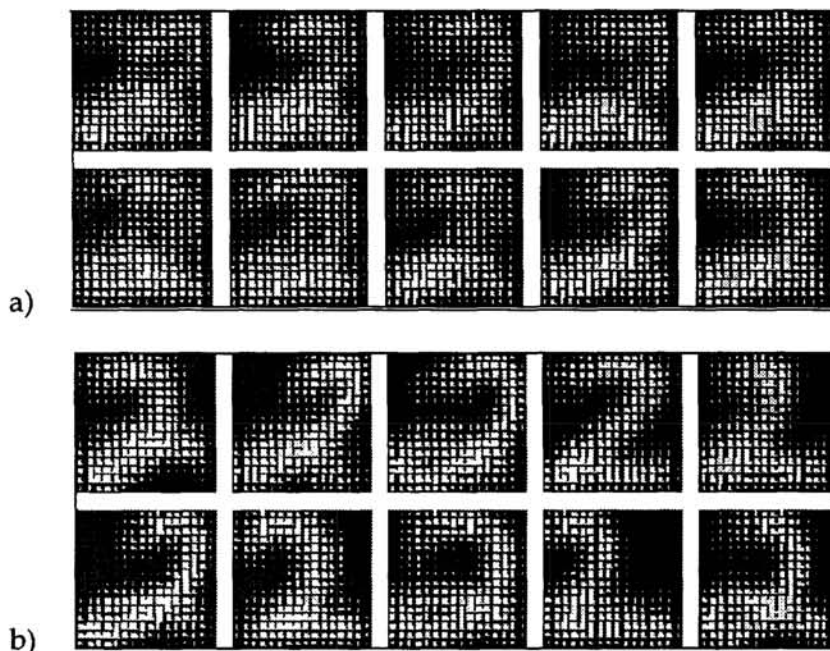

a)

b)

Figure 2: Cluster means for images of twos. (a) Without tangent vectors. (b) Cluster means using translation, rotation and scaling tangent vectors.

stochastically perturbed examples. The weightings applied to reconstruction of the tangent vectors are equivalent to the variances of the cloud.

There is an interesting relationship between mixtures of linear models and mixtures of Gaussians. Fitting a mixture of full-covariance Gaussians is typically infeasible when the dimensionality of the input image space, $n$, is high and the number of training examples is limited. Each Gaussian requires $n$ parameters to define its mean and $n(n + 1)/2$ more parameters to define its covariance matrix. One way to reduce the number of parameters is to use a diagonal covariance matrix but this is a very poor approximation for images because neighbouring pixel intensities are very highly correlated. An alternative way to simplify the covariance matrix is to flatten the Gaussian into a pancake that has $h$ dimensions within the pancake and $n - h$ dimensions orthogonal to the pancake. Within this orthogonal subspace we assume an isotropic Gaussian distribution (*ie* a diagonal covariance matrix with equal, small variances along the diagonal), so we eliminate $(n - h)(n - h + 1)/2$ degrees of freedom which is nearly all the degrees of freedom of the full covariance matrix if $h$ is small compared with $n$. Not counting the mean, this leaves $h(2n - h + 1)/2$ degrees of freedom which is exactly the number of parameters required to define the first $h$ principal components.[1] Thus we can view PCA as a way of fiercely constraining a full covariance Gaussian but nevertheless leaving it free to model

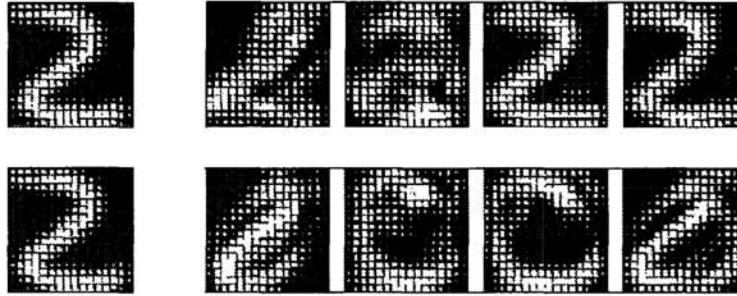

Figure 3: Reconstruction of a 2 (left column) using 2 (upper) and 0 (lower) models.

| Method | Raw Error Rate | Memory Requirements |
|---|---|---|
| Linear Models | 127 (4.68%) | 110 |
| Linear Models + Tangent Dist | 117 (4.3%) | 110 |
| Tangent Distance | 97 (3.58%) | 1100 |

Table 2: Classification errors on the official test test when no rejections are allowed. Memory requirements are indicated in terms of the number of $16 \times 16$ images that need to be stored.

important correlations.

We have investigated a number of extensions to the basic scheme. As only one of these yielded improved results on the validation set, we will only briefly review them. If instead of assuming an isotropic Gaussian within the pancake we use an ellipsoidal subspace, then we can can take into account the different variances along each of the h principal directions. This is akin to incorporating a *code cost* [14]. Similarly, the squared reconstruction error used in the basic scheme also assumes an isotropic distribution. Again a diagonal covariance matrix may be substituted. We surmise that this was not successful because we had insufficient training data to estimate the variances along some dimensions reliably; for example some of the edge pixels were never turned on in the training data for some models.

The Euclidean metric for the reconstruction error is convenient as it authorises the use of a powerful methods such as principal components; however as pointed out in the introduction it is deficient for situations like character recognition. We tried a scheme in which the models were trained as described above, but tangent distance [11] was used during testing. This method yielded marginally improved results on both the validation and test sets (Table 2). More adventurous tangent options, including using them in the clustering phase, were explored by [5, 10].

PCA models are not ideal as *generative* models of the data because they say nothing about how to generate values of components in the directions orthogonal to the pancake. To ameliorate this, the generative model may be formulated as $\mathbf{f}(\mathbf{p}) = A.\mathbf{p} + \epsilon$, where the components of $\epsilon$ are independent and the *factors* $\mathbf{p}$ have some prior covariance matrix. The generative weights of this autoencoder can

be obtained using the technique of maximum likelihood factor analysis (which is closely related to PCA) and the resulting architecture and hierarchical variants of it can be formulated as real valued versions of the Helmholtz machine [3, 6]. The cost of coding the factors relative to their prior is implicitly included in this formulation, as is the possibility that different input pixels are subject to different amounts of noise. Unlike PCA, factor analysis privileges the particular input coordinates (rather than being invariant to rotations of the input covariance matrix).

## Footnotes

[1]The $h^{\text{th}}$ principal component only requires $n - h + 1$ parameters because it must be orthogonal to all previous components.

[1]This research was funded by the Ontario Information Technology Research Centre and NSERC. We thank Patrice Simard, Chris Williams, Rob Tibshirani and Yann Le Cun for helpful discussions. Geoffrey Hinton is the Noranda Fellow of the Canadian Institute for Advanced Research.

# References

[1] Bourlard, H & Kamp, Y (1988). *Auto-association by Multilayer Perceptrons and Singular Value Decomposition*. Biol. Cybernetics 59, 291-294.

[2] Bregler, C & Omohundro, SM (1995). Non-linear image interpolation using surface learning. This volume.

[3] Dayan, P, Hinton, GE, Neal, RM & Zemel, RS (1995). The Helmholtz machine. *Neural Computation,* in press.

[4] Dempster, AP, Laird, NM & Rubin, DB (1976). Maximum likelihood from incomplete data via the EM algorithm. *Proceedings of the Royal Statistical Society,* 1–38.

[5] Hastie, T, Simard, P & Sackinger, E (1995). Learning prototype models for tangent distance. This volume.

[6] Hinton, GE, Dayan, P, Frey, BJ, Neal, RM (1995). The wake-sleep algorithm for unsupervised neural networks. Submitted for publication.

[7] Hinton, GE & Zemel, RS (1994). Autoencoders, minimum description length and Helmholtz free energy. In JD Cowan, G Tesauro & J Alspector, editors, *Advances in Neural Information Processing Systems 6.* San Mateo, CA: Morgan Kaufmann.

[8] Hull, JJ (1994). A database for handwritten text recognition research. *IEEE Transactions on Pattern Analysis and Machine Intelligence,* 16, 550-554.

[9] Kambhatla, N & Leen, TK (1994). Fast non-linear dimension reduction. In JD Cowan, G Tesauro & J Alspector, editors, *Advances in Neural Information Processing Systems 6.* San Mateo, CA: Morgan Kaufmann.

[10] Schwenk, H & Milgram, M (1995). Transformation invariant autoassociation with application to handwritten character recognition. This volume.

[11] Simard, P, Le Cun, Y & and Denker, J (1993). Efficient pattern recognition using a new transformation distance. In SJ Hanson, JD Cowan & CL Giles, editors, *Advances in Neural Information Processing Systems 5,* 50-58. San Mateo, CA: Morgan Kaufmann.

[12] Simard, P, Victorri, B, LeCun, Y & Denker, J (1992). Tangent Prop - A formalism for specifying selected invariances in an adaptive network. In JE Moody, SJ Hanson & RP Lippmann, editors, *Advances in Neural Information Processing Systems 4.* San Mateo, CA: Morgan Kaufmann.

[13] Williams, CKI, Zemel, RS & Mozer, MC (1993). Unsupervised learning of object models. In *AAAI Fall 1993 Symposium on Machine Learning in Computer Vision,* 20-24.

[14] Zemel, RS (1993). *A Minimum Description Length Framework for Unsupervised Learning.* PhD Dissertation, Computer Science, University of Toronto, Canada.

